# Learning Partially Observable Models Using Temporally Abstract Decision Trees

**Erik Talvitie**
Department of Mathematics and Computer Science
Franklin & Marshall College
Lancaster, PA 17604
`erik.talvitie@fandm.edu`

## Abstract

This paper introduces timeline trees, which are partial models of partially observable environments. Timeline trees are given some specific predictions to make and learn a decision tree over history. The main idea of timeline trees is to use temporally abstract features to identify and split on features of key events, spread arbitrarily far apart in the past (whereas previous decision-tree-based methods have been limited to a finite suffix of history). Experiments demonstrate that timeline trees can learn to make high quality predictions in complex, partially observable environments with high-dimensional observations (e.g. an arcade game).

## 1 Introduction

Learning a model of a high-dimensional environment can pose a significant challenge, but the ability to make predictions about future events is key to good decision making. One common approach is to avoid learning a complete, monolithic model of the environment, and to instead focus on learning *partial models* that are only capable of making a restricted set of predictions (for instance, predictions about some particular aspect of the environment, or predictions about future rewards). Partial models can often be simpler to learn than a complete model. In some cases they can be combined to form complete, structured models, which can then be used for planning purposes (e.g. factored MDPs [1], collections of partial models [2]). In other cases, partial models can be directly useful for control (e.g. U-Tree [3], prediction profile models [4]).

This paper introduces *timeline trees* which are partial models for partially observable environments. Timeline trees are focused on capturing a particular kind of partial observability; they assume that their predictions can be made by recalling a (finite) sequence of events in the past that may have occurred far apart from each other in time. While not all partially observable phenomena take this form, a good deal of everyday partial observability has this flavor. For instance, you may know that your keys are in the next room because you remember putting them there. Most of the experiences since that event are probably irrelevant to making predictions about the location of your keys.

The main idea of timeline trees is to build a decision tree over history. As with similar approaches, the decision tree can split on features of observations in recent history. However, a timeline tree may also establish new *timestamps* in the past and is able split on features of observations surrounding those events as well. For instance, there could be a timestamp representing the last time the agent saw its keys, and then features of the neighboring observations could identify the keys' location. In this way, timeline trees can make use of information arbitrarily spread out in history.

## 2 Partial Models

This paper will focus on discrete dynamical systems. Specifically, time procedes in discrete steps $t = 1, 2, 3, \ldots$. At every step $t$, the agent selects an *action* $a_t$ from a finite set $\mathcal{A}$ and the environment (stochastically) emits an *observation* $o_t$, taken from a finite set $\mathcal{O}$. The *history at time* $t$ is the sequence of actions and observations from the beginning of time, up through time $t$: $h_t \overset{\text{def}}{=} a_1 o_1 a_2 o_2 \ldots a_t o_t$. In the general partially observable case, the observation emitted at each step may depend upon the entire history (and the agent's action). So, an agent wishing to predict the next observation must model the conditional probability distribution $\Pr(O_{t+1} \mid H_t, A_{t+1})$.

If one is able to predict the next observation at any history and for any action (that is, if one has access to this conditional distribution), one can compute the probability of *any* future sequence of observations given any future sequence of actions and the history [5]. Such a model is called a *complete model* because in any situation, it is capable of making any prediction about the future. Examples in the partially observable setting include POMDPs [6, 7] and PSRs [5, 8]. A *partial model* is any model that does not represent this full conditional distribution.

This paper will focus on partial models that make conditional predictions about abstract features of the next observation, though many of the ideas can be straightforwardly adapted to work with predictions of other forms. Formally, let $\omega$ and $\kappa$ be many-to-one mappings over the set of observations $\mathcal{O}$. The task of the partial model at time $t$ will be to predict the value of $\omega(o_{t+1})$, conditioned on the value of $\kappa(o_{t+1})$. So it represents the distribution $\Pr(\omega(O_{t+1}) \mid H_t, A_{t+1}, \kappa(O_{t+1}))$. For example, in the experiment in Section 5.3, observations are images and multiple partial models are learned, each predicting the color of a single pixel, conditioned on the colors of pixels above and to the left.

### 2.1 Related Work: Partial Models in Partially Observable Environments

McCallum's U-Tree [3] learns a decision tree over history, where the leaves of the tree map to the expected discounted sum of rewards at the associated history (though the method could be adapted to make other predictions, as in [9]). McCallum used binary features of the form "Feature $X$ takes value $Y$ at time-step $t - k$," where $t$ is the current time-step. Thus the decision tree learns an abstraction *both* over observations (which could be high-dimensional in their own right) *and* over the sequence of observations (by using features from multiple time-steps). However, because it can only consider a finite number of such features, U-Tree has a finite memory horizon; events that occur before some arbitrary cutoff in the past cannot be taken into account when making predictions.

Timeline trees are an extension of UTree that allow it to use features of observations arbitrarily far back in the past, though they are not the first attempt to address this issue. Looping predictive suffix trees (LPSTs) [10] are prediction suffix trees [11] that allow nodes to loop back to their ancestors. Local agent state representations (LASR) [12] map histories to a real number, and then learn a direct mapping from that number to the target predictions. McCallum [13] and Mahmud [14] both developed incremental hill-climbing algorithms to learn finite state machines (FSMs), where each state is associated with predictions about future rewards, and the transitions depend on both the action taken and the observation received. Prediction profile models [4] are similar FSMs, but rather than hill-climbing, they are learned by pre-processing the data and then applying standard complete model learning methods (they were demonstrated using POMDPs and LPSTs).

All of these approaches can, in principle, represent arbitrarily long-range dependencies in time. However, unlike U-Tree, they *all* treat observations as atomic, which limits their applicability to truly high-dimensional systems. Furthermore, despite their theoretical representational capacity, their learning algorithms have difficulty discovering long-range dependencies in practice.

The learning algorithm for LPSTs first learns a full suffix tree, and then adds loops as appropriate. Thus, to capture very long-range dependencies, one must first build a very deep suffix tree. McCallum reported that his FSM-learning method was often unable to detect long-range temporal dependencies (since this typically involves multiple elaborations of the FSM, none of which would individually seem valuable to the hill-climbing algorithm). Mahmud's similar approach would likely suffer a similar limitation. The learning algorithms for LASR and prediction profile models both rely on estimating predictions at particular histories. Because estimates will only be accurate for histories that appear many times, these algorithms can only be effectively applied to data consisting of many short trajectories, which limits their ability to discover long-range dependencies in practice.

It should be noted that prediction profile models have been combined with an additional pre-processing step that learns an abstraction before the prediction profile model is learned [15]. Because of this, and because their formulation most closely fits the setting of this paper, the experiments in Section 5 will directly compare against the performance of prediction profile models.

## 3   Timeline Trees

The goal of timeline trees is to combine the strengths of U-Tree with the ability to attend to events arbitrarily far apart in history (rather than limited to a finite suffix). Unlike several of the above approaches, timeline trees are not arbitrarily recurrent (they do not contain loops except in a limited, implicit sense), which does restrict their representational capacity. However, in exchange they retain the straightfoward decision tree training of U-Tree, which allows them to simultaneously learn an abstraction over both the history sequence *and* high-dimensional observations and which further-more allows them to discover long-range temporal depencies in practice (and not just in principle).

### 3.1   Timestamps

The decision tree built by U-Tree splits on features of observations at some temporal offset from the current timestep. The key idea of timeline trees is to allow multiple *timestamps* in history and to allow splits on features of observations at temporal offsets relative to any of these timestamps. Timeline trees take a set $\mathcal{F}$ of binary features, where each feature $f(h_t, k)$ takes the history at time $t$, $h_t$, and a timestep $0 < k \leq t + 1$[1] and returns 1 or 0. For example, if the observations are images, $f$ could return 1 if a black pixel existed anywhere at step $k - 1$ but did not exist at step $k$. It is assumed that $f(h_t, k)$ makes use of no information *after* step $k$ (though it may access timestep $k$ or before).

For a fixed vector $\tau$ of timestamps, the model is a standard decision tree, and only a small extension of U-Tree (which fixed the number of timestamps to 1: the current timestep). Each internal node in the tree is associated with a feature $f$, a timestamp index $i$, and a temporal offset $\delta$ (which may be negative) and has two children representing histories where the value of $f(h_t, \tau[i] + \delta)$ is 0 or 1, re-spectively. The leaves of the tree are associated with estimates of $\Pr(\omega(O_{t+1}) \mid h_t, a_{t+1}, \kappa(O_{t+1}))$. To use the timeline tree to make a prediction, one simply follows a path from the root to a leaf in the tree, choosing the appropriate child at each node according to the feature value $f(h_t, \tau[i] + \delta)$.

Timeline trees' real strength lies in their ability to *add* new timestamps. They do this via a special type of feature. For every feature $f \in \mathcal{F}$, there is an additional *timestamp feature* $\xi^f$. The feature $\xi^f(h_t, j, k) = 1$ if there is some timestep $m$ such that $j < m < k$ where $f(h_t, m) = 1$. More importantly, the greatest such $m$ (that is, the time of the most recent occurence of $f$), call it $m_f$, is added as a timestamp to all nodes in the subtree where $\xi^f = 1$.

When making a prediction for $h_t$, one maintains a growing vector $\tau$ of timestamps (in order of least to most recent). Beginning at the root there is only one timestamp: $\tau_{root} = \langle t + 1 \rangle$, where $t$ is the current timestep. As one travels from the root to a leaf, one may encounter a node associated with timestamp feature $\xi^f$. Such a node is also associated with an index $i$ into the *current* timestamp vector. If $\xi^f(h_t, \tau[i - 1], \tau[i]) = 1$, the path moves to the corresponding child *and* adds $m_f$ to $\tau$ (let $\tau[0] \stackrel{\text{def}}{=} -1$). Nodes further down in the tree may refer to this new timestamp. As such, the tree is able to establish timestamps based on the occurence of key events (the presence of some feature).

Timestamp features are a form of temporal abstraction; they refer to an event in the past, but abstract away how long ago it was. They are limited, however. There are systems that would require an infi-nite timeline tree that approaches in Section 2.1 can capture easily (see Section 5.2). Nevertheless, they do capture a natural and intuitive form of partial observability, as can be seen by example.

### 3.2   An Example

As a simple illustrative example, consider an agent that must keep track of its key. The agent's key can be in room $A$ or $B$, or in the agent's pocket (where it starts). The agent has three actions: *move*

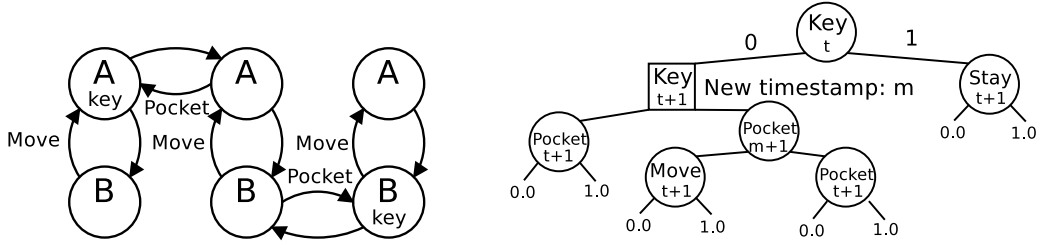

Figure 1: The Key World example.

(which switches its location), *stay* (which does nothing), and *pocket*. The last action transfers the key between the agent's pocket and the current room (in either direction) unless the key is in neither (in which case it does nothing). The agent can observe its location and whether the key is in the room. A diagram is shown in the left of Figure 1 (missing arrows are self-loops).

On the right of Figure 1 an example timeline tree is shown that can predict whether the agent will see the key in the next timestep. At the root, there is only one timestamp: $t+1$, where $t$ is the current step. The root checks if the agent can currently see the key. If so, the agent will only see the key in the next step if it *stay*s. Otherwise, the agent must remember where the key was last seen. The square-shaped node is meant to indicate a timestamp feature, which checks if the agent has *ever* seen the key before the only timestamp. If not, the key is in the agent's pocket. If so, a *new* timestamp $m$ is added that marks the last time the key was seen. If the agent put the key in its pocket after $m$, it must take the key out to see it. Otherwise, it must be in the other room.

## 4 Learning Timeline Trees

Timeline trees can be learned using standard decision tree induction algorithms (e.g. ID3 [16] and C4.5 [17]). The leaves of the tree contain the estimated predictions (counts of the occurrences of the various values of $\omega(o)$ associated with histories mapping to that leaf). The tree starts as just the root (not associated with a feature). Each phase of training expands a single leaf by associating it with a feature and adding the appropriate children under it. At each phase every candidate expansion (every leaf and every feature) is tried and the one that results in the highest information gain between the predictions of the original tree and the expanded tree is greedily selected.

The main difference in timeline trees is that different features may be available in different leaf nodes (because different timestamps will be available). Specifically for each leaf $n$, all features of the form $f(\cdot, \tau_n[i] + k)$ are considered for all timestamp indices $i \in \{1, \ldots, |\tau_n|\}$ and all integer offsets $k$ in some finite range. Similarly, all timestamp features of the form $\xi^f(\cdot, \tau_n[i-1], \tau_n[i])$ are considered for all timestamp indices $i$. In the experiments below, candidate expansions also include all *combinations* of timestamp features and regular features (essentially two expansions at once). These compound features take the form of first splitting on a timestamp feature, and then splitting the resulting "1 child" with a regular feature. This allows the tree to notice that a timestamp is useful for the subsequent splits it allows, even if it is not inherently informative itself. For instance, in the Key World, knowing whether the agent has ever seen the key may not be very informative, but knowing that the *pocket* action was taken immediately after seeing the key is very informative.

Note that compound features will tend to result in higher information gain than simple features. As a result, there will be a bias toward selecting compound features, which is not necessarily desireable. To combat this, the information gain of compound features was penalized by a factor of $\beta$. In the experiments below, $\beta = 0.5$. Also note that because the information gain measurement used to choose expansions is estimated from a finite number of samples, expanding the tree until information gain is zero for all candidates will typically result in overfitting. Thus, some form of early stopping is common. In this implementation expansions are only considered if they make a statistically significant change to the predictions (as measured by a likelihood ratio test). The statistical test requires a significance level, $\alpha$, which controls the probability of detecting a spurious difference. Applying the test several times to the same data set compounds the danger of such an error, so $\alpha$ should be set quite low. In the experiments below, $\alpha = 10^{-10}$.

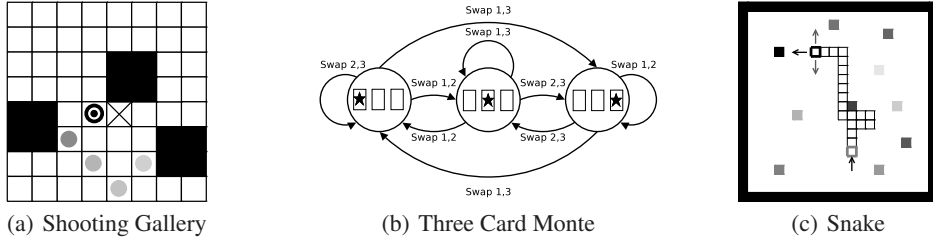

(a) Shooting Gallery       (b) Three Card Monte       (c) Snake

Figure 2: Experiment Domains

# 5  Experiments

In this section, timeline trees will be evaluated in three problems to which prediction profile models have been previously applied. In each problem a set of features and a set of training trajectories are provided. For various amounts of training trajectories, timeline trees are learned and their prediction accuracy is evaluated (as well as their usefulness for control). Results are averaged over 20 trials. Note that for prediction profile models a completely new model is learned for each batch of training trajectories. For timeline trees, the new data is simply added to the existing tree and new splits are made until the algorithm stops. This strategy is effective for timeline trees since the initial splits can often be made with relatively little data (this not possible for prediction profile models).

In addition to evaluating timeline trees, two variants will also be evaluated. One (labeled "Finite Suffix") does not use any timestamp features at all. Thus, it is similar to U-Tree (splitting on features of a finite suffix of history). The other (labeled "No Timestamps") includes timestamp features, but does not use them to create new timestamps. This variant is meant to evaluate whether any performance benefit is due to the form of the features or due to the addition of new timestamps.

## 5.1  Shooting Gallery

In this example, from Talvitie and Singh [4], the agent is in a shooting gallery (see Figure 2(a)). Its gun is aimed at a fixed position (marked by the "X") and it must shoot a target that moves around the grid, bouncing off the edges and obstacles (an example trajectory is pictured). If the target is in the crosshairs in the step *after* the agent shoots, the agent gets a reward of 10. Otherwise it gets a reward of -5. Whenever the agent hits the target, the gallery resets (obstacles are placed randomly) and an special observation is emitted. The gallery may also reset on its own with a 0.01 chance.

Clearly the agent must predict whether the target will be in the crosshairs in the next timestep, but the target's movement is stochastic and partially observable. At every step it either moves in its current direction with probability 0.7 or stays in place with probability 0.3. The agent must remember the direction of the ball the *last time it moved*. This problem is also fairly high-dimensional. There are roughly 4,000,000 possible observations, and even more hidden states. Because of the large number of observations Talvitie and Singh [4] hand-crafted an observation abstraction and applied it to the training data before learning the prediction profile models. Their abstraction pays attention only to the position of the target and the configuration of the obstacles in its immediate neighborhood. By constrast, timeline trees *learn* an abstraction over both observations and the history sequence.

**Experimental Setup:** The prediction profile models were trained on trajectories of length 4, generated by the uniform random policy. Though short trajectories are necessary for training prediction profile models, the timeline trees tended to overfit to the short trajectories. In short trajectories, a feature like "Has the target ever been in the crosshairs?" might seem spuriously meaningful. During testing, which takes place on one long trajectory, this feature would be much less informative. Therefore, the tree models were trained on fewer, longer trajectories (of length 40).

To train the tree models, a binary feature was provided for each color (target, obstacle, background, or reset) for each pixel in the image. There was also a feature for each action. The maximum temporal offset from a timestamp was set to 2.

The learned models are evaluated by using their predictions as features for a policy gradient algorithm, OLGARB [18]. Good predictions about the color under the cross-hairs should lead to a good policy. For the details of how the predictions are encoded for OLGARB, see [4]. To evaluate the learned models, OLGARB is run for 1,000,000 steps. The average reward obtained and the root

mean squared error (RMSE) of the probabilities provided by the model are reported (at each step, the model's probability that the target will be in the crosshairs is compared to the true probability).

**Results:** Figure 3 shows the results. The line marked "Prediction Profile" shows the best results reported by Talvitie and Singh [4]; the other curves show the performance of timeline trees and the comparison variants. In the control performance graph, the dashed line marked "Optimal" shows the average performance of the optimal policy. The dashed line marked "True" shows the average performance of OLGARB when given the true predictions as features. This is the best performance a learned model could hope for.

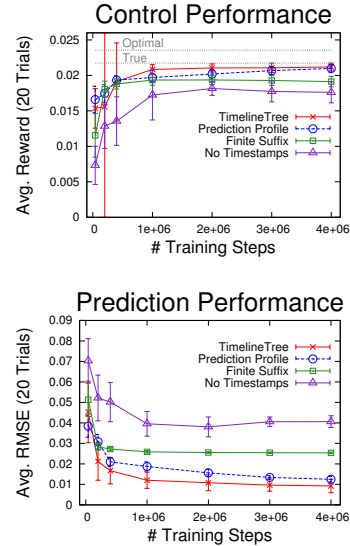

Both the timeline trees and the prediction profile models are able to learn to make good predictions, but timeline trees do so with less data. Remember that timeline trees are learning from raw images whereas the prediction profile models have been provided a hand-crafted abstraction. The tree models without timestamps are only able to make good predictions in histories where the target has recently moved, which limits their performance. The "No Timestamp" variant is outperformed by the "Finite Suffix" model, which indicates that, despite the longer training trajectories, it may still be overfitting.

Figure 3: Shooting gallery results.

## 5.2 Three Card Monte

The next example, also from Talvitie and Singh [4], is one for which the decision tree approach would not be appropriate. While illustrating the limitations of timeline trees in comparison to more expressive methods, it also demonstrates that they can represent useful knowledge that the simpler tree-based methods cannot. The problem is based on the simple game "Three Card Monte". There are three face down cards on the table, one of which is the ace. A dealer repeatedly chooses two cards and swaps their positions. Eventually the dealer asks the agent to flip over the ace. If the agent succeeds, it gets a reward of 1; if it fails it gets a reward of -1. For a detailed specification, see [4].

Note that to do well in this game, the agent only needs to make the prediction, "If I flip card 1, will it be the ace?" (and the corresponding predictions for the other 2 cards) at any history. It does not, for instance, need to predict which cards will be swapped in the next time step. A complete model would attempt to make this prediction, which would mean not only modeling the movement of the cards, but also the decision making process of the dealer! The dealer in these experiments choses the swap it has chosen *least frequently* so far with probability 0.5. With probability 0.4 it choses uniformly randomly between the other two swaps. With probability 0.1, it asks for a guess. Since modeling the dealer's behavior requires counting the number of times each swap has been selected, a complete POMDP model of this system would require infinitely many states.

Further note that the *entire* sequence of swap observations since the last time the ace's position was observed is important for predicting the ace's location. Since timeline trees' primary strength is ignoring sections of history to focus on a few key events, they would not be expected to model this problem well. Prediction profile models, on the other hand, are able to track the ace's location with a 3-state machine (pictured in Figure 2(b)).

**Experimental Setup:** Training and evaluation were the same as in the Shooting Gallery (above) except the prediction profile models were given length 10 trajectories and the tree-based models were given length 100 trajectories. The features provided to the trees were encodings of the atomic actions and observations. There was a binary feature indicating each observation, action, and each action-observation pair. The maximum time offset from a timestamp was 10 steps (both positive and negative). The specification of the prediction profile models implicitly encodes the fact that the agent's action $a_{t+1}$ is important to the predictions (i.e. which card it flips). For fairness, the tree models were also seeded with these features (they were split on the agent's action before training).

**Results:** Figure 4(a) presents the control performance results and Figure 4(b) shows the prediction error results. The prediction profile models are able to perfectly track the ace's location after 100,000

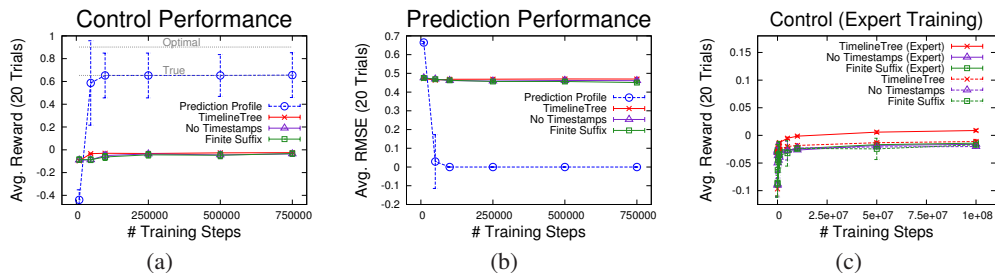

Figure 4: Results in Three Card Monte.

training steps. As expected, the tree methods perform poorly (negative average reward indicates more wrong guesses than right), though timeline trees have marginally better control performance.

Part of the difficulty is that randomly generated training data is quite different than what the agent will encounter during testing (the random agent flips cards over frequently, while the learning agent eventually flips a card only when prompted to). Figure 4(c) shows the control performance of the tree models trained with *expert*-generated data instead (generated by the optimal policy). The dashed lines show the results for random training data for comparison. Expert-trained timeline trees are eventually good enough to allow the agent to achieve positive average reward, though they do require a great deal of data to do so (note the changes to the axes). Though the expert training improves the performance of the limited variants as well, neither achieves positive average reward. So, though their representational limitations do prevent all three tree-based methods from performing well in this problem, timeline trees' ability to create new timestamps seems to allow them to make some meaningful (and useful) predictions that the others cannot.

## 5.3 Snake

The final example is an arcade game called "Snake" from Talvitie [15] (see Figure 2(c)). In this problem, multiple partial models will be learned and combined to form a complete, structured model which can be used for planning. The agent controls the head of a snake. The snake's body trails behind, and the tail does exactly what the head did, at a delay. There are 10 food pellets on the screen and the goal is to eat them in a particular order (indicated by the shades of grey in Figure 2(c)). If the snake ever runs into the wrong pellet, its own body, or the edge of the screen, the game is over and the agent receives -0.01 reward. Whenever the snake eats a pellet, the agent gets 1 reward and the tail stays still for 5 timesteps, making the snake's body longer. In addition, there is a 0.2 chance each step that the tail will not move, so the snake is always growing, imposing some time-pressure.

This version of Snake has two sources of partial observability. The tail shadows the head and, in addition, the pellet the snake must eat next is invisible. Initially, all 10 pellets are shown, but when the first pellet is eaten, the next one disappears, and only reappears if the snake's head is adjacent to it. When that pellet is eaten, the next pellet disappears, and so on. To do well, the agent must remember the location of the next pellet. The observations are $20 \times 20$ images. There are over $10^{30}$ distinct possible observations and even more hidden states.

**Experimental Setup:** For every location $(x, y)$ and every color $c$, a timeline tree model was used to predict whether the pixel at $(x, y)$ would next be color $c$. These models jointly predict the entire next observation. The models were ordered, and each model could condition on outcomes predicted by the previous models (so $\kappa$ gave the portion of the image predicted by models earlier in the order). In this case, the models were ordered by color, then by location (column-major order). The color order was head, tail, body, the 10 food pellet colors (in increasing order), and finally background.

For each color, training data was pooled data across all locations (rather than learn a separate model for each coordinate). However, features were provided that gave the position of the model (there was a binary feature for each column and each row), so the tree could choose to attend to position information if necessary. There was also a binary feature for each action and several pixel-based features. There was a feature for each color of each pixel in a $5 \times 5$ square around the pixel being predicted. There were also features for the same square of pixels indicating whether each pixel had just changed from one color to another color (for all pairs of colors). Finally, there were features indicating whether a particular color or particular color change existed anywhere in the image.

Prediction profile models were applied to this problem by Talvitie [15]. Similarly, multiple prediction profile models were learned, each responsible for a particular prediction *in particular situations* (called "histories of interest"). For instance, one model type predicted whether a particular pixel would contain the head in the next timestep, but only when the head was in the immediate neighborhood of that pixel. Before training the prediction profile models, an abstraction learning process was applied to the data that mapped each action-observation sequence between histories of interest to a single abstract observation. Thus, even though the raw data might contain very long trajectories, the abstract data consisted of only short trajectories. The reader is referred to Talvitie [15] for a detailed description of this approach. The main thing to note is that the hand-crafted structure indicates to each model which key events it should attend to, and which stretches of history can be ignored. By contrast, timeline trees *learn* this information.

As in [15], the training data was generated by running UCT [19] (a sample-based planning algorithm) on the true model (with a 0.25 chance of taking a random action). Each training trajectory is a full game (typically a few hundred steps long). The learned set of partial models was evaluated collectively. The joint model was given to UCT and its average planning performance over 100 test games was compared to that of UCT with a perfect model. The probability given by the model for the observation at each step was compared to the true probability and the RMSE has been reported.

**Results:** The results are shown in Figure 5. Despite the hand-crafted structure provided to the prediction profile models, timeline trees learn higher quality models with less data. In fact, UCT appears to perform better using the timeline tree model than the true model (marked "True"). This is due to a coincidental interaction between the model and UCT. The learned model mistakenly predicts that the snake may not die when it moves off the edge of the screen (a rare event in the training data). This emboldens UCT to consider staying near the edges, which can be necessary to escape a tight spot.

In terms of control performance, the "No Timestamps" variant performs nearly identically to the full timeline tree. This is because they are equally good at tracking the invisible food pellet. The model checks if each pixel has ever contained a food pellet and if it has ever contained the head. If "yes" and "no," respectively then it must contain food. This can be expressed without creating new timestamps. However, the "No Timestamps" model cannot fully represent a model of the tail's movement (which requires remembering what the head did when it was in the tail's position). The timeline tree incurs substantially less prediction error, indicating that it is able to model the tail more accurately.

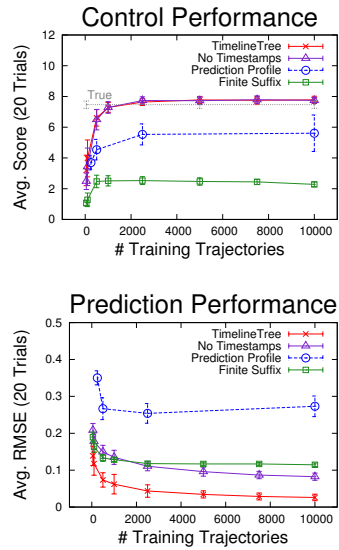

Figure 5: Results in Snake.

## 6 Conclusions

In these experiments, timeline trees learned to capture long-range dependencies in complex, partially observable systems with high-dimensional observations. The assumption that the predictions of interest depend on only a few key events in the past is limiting in the sense that there are simple partial models that timeline trees cannot easily capture (e.g. Three Card Monte), but it does reflect a broad, natural class of partially observable phenomena (the examples here, for instance, were not designed with timeline trees in mind). In problems that do match timeline trees' inductive biases, they have been shown to outperform the more expressive prediction profile models.

There are many possible directions in which to consider extending timeline trees. More sophisticated decision tree induction methods could help with sample complexity and overfitting. Regression tree methods could extend timeline trees into environments with continuous dimensions. The timestamp features used here are only one of many possible types of temporally abstract features that could be devised. Of particular interest is whether the ideas here can be combined with approaches described in Section 2.1 in order to increase expressive power, while retaining the benefits of timeline trees.

## Footnotes

[1]For simplicity's sake, the notation $f(h_t, k)$ hides the fact that the features may also depend on $a_{t+1}$ and $\kappa(o_{t+1})$. For this discussion, assume that $k$ may equal $t + 1$ if the feature makes use of *only* these aspects of time $t + 1$. If $k = t + 1$ and the feature refers to other information, assume $f(h_t, k) = 0$.

## References

[1] Craig Boutilier, Thomas Dean, and Steve Hanks. Decision-theoretic planning: Structural assumptions and computational leverage. *Journal of Artificial Intelligence Research*, 11:1–94, 1999.

[2] Erik Talvitie and Satinder Singh. Simple local models for complex dynamical systems. In *Advances in Neural Information Processing Systems 21 (NIPS)*, pages 1617–1624, 2009.

[3] Andrew K. McCallum. *Reinforcement Learning with Selective Perception and Hidden State*. PhD thesis, Rutgers University, 1995.

[4] Erik Talvitie and Satinder Singh. Learning to make predictions in partially observable environments without a generative model. *Journal of Artificial Intelligence Research (JAIR)*, 42:353–392, 2011.

[5] Michael Littman, Richard Sutton, and Satinder Singh. Predictive representations of state. In *Advances in Neural Information Processing Systems 14 (NIPS)*, pages 1555–1561, 2002.

[6] George E. Monahan. A survey of partially observable markov decisions processes: Theory, models, and algorithms. *Management Science*, 28(1):1–16, 1982.

[7] Anthony R. Cassandra, Leslie Pack Kaelbling, and Michael L. Littman. Acting optimally in partially observable stochastic domains. In *Proceedings of the Twelfth National Conference on Artificial Intelligence (AAAI)*, volume 2, pages 1023–1028, 1994.

[8] Satinder Singh, Michael R. James, and Matthew R. Rudary. Predictive state representations: A new theory for modeling dynamical systems. In *Uncertainty in Artificial Intelligence: Proceedings of the Twentieth Conference (UAI)*, pages 512–519, 2004.

[9] Alicia Peregrin Wolfe and Andrew G. Barto. Decision tree methods for finding reusable MDP homomorphisms. In *Proceedings of the Twenty-First National Conference on Artificial Intelligence (AAAI)*, 2006.

[10] Michael Holmes and Charles Isbell. Looping suffix tree-based inference of partially observable hidden state. In *Proceedings of the Twenty-Third International Conference on Machine Learning (ICML)*, pages 409–416, 2006.

[11] Dana Ron, Yoram Singer, and Naftali Tishby. The power of amnesia. In *Advances in Neural Information Processing Systems 6*, pages 176–183, 1994.

[12] Monica Dinculescu and Doina Precup. Approximate predictive representations of partially observable systems. In *Proceedings of the Twenty-Seventh International Conference on Machine Learning (ICML)*, pages 895–902, 2010.

[13] R. Andrew McCallum. Overcoming incomplete perception with utile distinction memory. In *Proceedings of the Tenth International Conference on Machine Learning (ICML)*, pages 190–196, 1993.

[14] M. M. Hassan Mahmud. Constructing states for reinforcement learning. In *Proceedings of the Twenty-Seventh International Conference on Machine Learning (ICML)*, pages 727–734, 2010.

[15] Erik Talvitie. *Simple Partial Models for Complex Dynamical Systems*. PhD thesis, University of Michigan, Ann Arbor, MI, 2010.

[16] J. Ross Quinlan. Induction of decision trees. *Machine Learning*, 1:81–106, 1986.

[17] J. Ross Quinlan. *C4.5: Programs for Machine Learning*. Morgan Kaufman Publishers Inc., San Francisco, CA, 1993.

[18] Lex Weaver and Nigel Tao. The optimal reward baseline for gradient-based reinforcement learning. In *Uncertainty in Artificial Intelligence: Proceedings of the Seventeenth Conference (UAI)*, pages 538–545, 2001.

[19] Levente Kocsis and Csaba Szepesvári. Bandit based monte-carlo planning. In *Proceedings of the Seventeenth European Conference on Machine Learning (ECML)*, pages 282–293, 2006.

